# Multiple Choice Learning: Learning to Produce Multiple Structured Outputs

**Abner Guzman-Rivera**
University of Illinois
aguzman5@illinois.edu

**Dhruv Batra**
Virginia Tech
dbatra@vt.edu

**Pushmeet Kohli**
Microsoft Research Cambridge
pkohli@microsoft.com

## Abstract

We address the problem of generating multiple hypotheses for structured prediction tasks that involve interaction with users or successive components in a cascaded architecture. Given a set of multiple hypotheses, such components/users typically have the ability to retrieve the best (or approximately the best) solution in this set. The standard approach for handling such a scenario is to first learn a single-output model and then produce $M$-Best Maximum a Posteriori (MAP) hypotheses from this model. In contrast, we *learn to produce multiple outputs* by formulating this task as a multiple-output structured-output prediction problem with a loss-function that effectively captures the setup of the problem. We present a max-margin formulation that minimizes an upper-bound on this loss-function. Experimental results on image segmentation and protein side-chain prediction show that our method outperforms conventional approaches used for this type of scenario and leads to substantial improvements in prediction accuracy.

## 1 Introduction

A number of problems in Computer Vision, Natural Language Processing and Computational Biology involve predictions over complex but structured interdependent outputs, also known as *structured-output prediction*. Formulations such as Conditional Random Fields (CRFs) [18], Max-Margin Markov Networks ($M^3N$) [27], and Structured Support Vector Machines (SSVMs) [28] have provided principled techniques for learning such models.

In all these (supervised) settings, the learning algorithm typically has access to input-output pairs: $\{(x_i, y_i) \mid x_i \in \mathcal{X}, y_i \in \mathcal{Y}\}$ and the goal is to learn a mapping from the input space to the output space $f : \mathcal{X} \to \mathcal{Y}$ that minimizes a (regularized) task-dependent loss function $\ell : \mathcal{Y} \times \mathcal{Y} \to \mathbb{R}^+$, where $\ell(y_i, \hat{y}_i)$ denotes the cost of predicting $\hat{y}_i$ when the correct label is $y_i$.

Notice that the algorithm always makes a *single* prediction $\hat{y}_i$ and pays a penalty $\ell(y_i, \hat{y}_i)$ for that prediction. However, in a number of settings, it might be beneficial (even necessary) to make *multiple* predictions:

1. **Interactive Intelligent Systems.** The goal of interactive machine-learning algorithms is to produce an output for an expert or a user in the loop. Popular examples include tools for interactive image segmentation (where the system produces a cutout of an object from a picture [5, 25]), systems for image processing/manipulation tasks such as image denoising and deblurring (*e.g.*, Photoshop), or machine translation services (*e.g.*, Google Translate). These problems are typically modeled using structured probabilistic models and involve computing the Maximum a Posteriori (MAP) solution. In order to minimize user interactions, the interface could show not just a single prediction but a small set of diverse predictions, and simply let the user pick the best one.

2. **Generating $M$-Best Hypotheses.** Machine learning algorithms are often cascaded, with the output of one model being fed into another. In such a setting, at the initial stages it is

not necessary to make the perfect prediction, rather the goal is to make a set of *plausible* predictions, which may then be re-ranked or combined by a secondary mechanism. For instance, in Computer Vision, this is the case for state-of-the-art methods for human-pose estimation which produce multiple predictions that are then refined by employing a temporal model [23, 3]. In Natural Language Processing, this is the case for sentence parsing [8] and machine translation [26], where an initial system produces a list of $M$-Best hypotheses [12, 24] (also called k-best lists in the NLP literature), which are then re-ranked.

The common principle in both scenarios is that we need to generate a set of plausible hypotheses for an algorithm/expert downstream to evaluate. Traditionally, this is accomplished by learning a single-output model and then producing $M$-Best hypotheses from it (also called the $M$-Best MAP problem [20, 11, 29] or the Diverse $M$-Best problem [3] in the context of graphical models).

Notice that the single-output model is typically trained in the standard way, *i.e.*, either to match the data distribution (max-likelihood) or to score ground-truth the highest by a margin (max-margin). Thus, there is a disparity between the way this model is trained and the way it is actually used. The key motivating question for this paper is – can we *learn to produce a set of plausible hypotheses*? We refer to such a setting as *Multiple Choice Learning* (MCL) because the learner must learn to produce multiple choices for an expert or other algorithm.

**Overview.** This paper presents an algorithm for MCL, formulated as multiple-output structured-output learning, where given an input sample $x_i$ the algorithm produces a set of $M$ hypotheses $\{\hat{y}_i^1, \ldots, \hat{y}_i^M\}$. We first present a meaningful loss function for this task that effectively captures the setup of the problem. Next, we present a max-margin formulation for training this $M$-tuple predictor that minimizes an upper-bound on the loss-function. Despite the popularity of $M$-Best approaches, to the best our knowledge, this is the first attempt to directly model the $M$-Best prediction problem. Our approach has natural connections to SSVMs with latent variables, and resembles a structured-output version of k-means clustering. Experimental results on the problems of image segmentation and protein side-chain prediction show that our method outperforms conventional $M$-Best prediction approaches used for this scenario and leads to substantial improvements in prediction accuracy.

The outline for the rest of this paper is as follows: Section 2 provides the notation and discusses classical (single-output) structured-output learning; Section 3 introduces the natural task loss for multiple-output prediction and presents our learning algorithm; Section 4 discusses related work; Section 5 compares our algorithm to other approaches experimentally and; we conclude in Section 6 with a summary and ideas for future work.

## 2   Preliminaries: (Single-Output) Structured-Output Prediction

We begin by reviewing classical (single-output) structured-output prediction and establishing the notation used in the paper.

**Notation.** For any positive integer $n$, let $[n]$ be shorthand for the set $\{1, 2, \ldots, n\}$. Given a training dataset of input-output pairs $\{(x_i, y_i) \mid i \in [n], x_i \in \mathcal{X}, y_i \in \mathcal{Y}\}$, we are interested in learning a mapping $f : \mathcal{X} \to \mathcal{Y}$ from an input space $\mathcal{X}$ to a structured output space $\mathcal{Y}$ that is finite but typically exponentially large (*e.g.*, the set of all segmentations of an image, or all English translations of a Chinese sentence).

**Structured Support Vector Machines (SSVMs).** In an SSVM setting, the mapping is defined as $f(x) = \operatorname{argmax}_{y \in \mathcal{Y}} \mathbf{w}^T \phi(x, y)$, where $\phi(x, y)$ is a joint feature map: $\phi : \mathcal{X} \times \mathcal{Y} \to \mathbb{R}^d$. The quality of the prediction $\hat{y}_i = f(x_i)$ is measured by a task-specific loss function $\ell : \mathcal{Y} \times \mathcal{Y} \to \mathbb{R}^+$, where $\ell(y_i, \hat{y}_i)$ denotes the cost of predicting $\hat{y}_i$ when the correct label is $y_i$. Some examples of loss functions are the intersection/union criteria used by the PASCAL Visual Object Category Segmentation Challenge [10], and the BLEU score used to evaluate machine translations [22].

The task-loss is typically non-convex and non-continuous in $\mathbf{w}$. Tsochantaridis *et al.* [28] proposed to optimize a regularized surrogate loss function:

$$\min_{\mathbf{w}} \quad \frac{1}{2} \|\mathbf{w}\|_2^2 + C \sum_{i \in [n]} \hbar_i(\mathbf{w}) \tag{1}$$

where C is a positive multiplier and $\hbar_i(\cdot)$ is the structured hinge-loss:

$$\hbar_i(\mathbf{w}) = \max_y \left( \ell(y_i, y) + \mathbf{w}^T \boldsymbol{\phi}(x_i, y) \right) - \mathbf{w}^T \boldsymbol{\phi}(x_i, y_i). \tag{2}$$

It can be shown [28] that the hinge-loss is an upper-bound on the task loss, *i.e.*, $\hbar_i(\mathbf{w}) \geq \ell(y_i, f(x_i))$. Moreover, $\hbar_i(\mathbf{w})$ is a non-smooth convex function, and can be equivalently expressed with a set of constraints:

$$\min_{\mathbf{w}, \xi_i} \quad \frac{1}{2} \|\mathbf{w}\|_2^2 + C \sum_{i \in [n]} \xi_i \tag{3a}$$

$$s.t. \quad \mathbf{w}^T \boldsymbol{\phi}(x_i, y_i) - \mathbf{w}^T \boldsymbol{\phi}(x_i, y) \geq \ell(y_i, y) - \xi_i \qquad \forall y \in \mathcal{Y} \setminus y_i \tag{3b}$$

$$\xi_i \geq 0 \tag{3c}$$

This formulation is known as the margin-rescaled n-slack SSVM [28]. Intuitively, we can see that it minimizes the squared-norm of $\mathbf{w}$ subject to constraints that enforce a soft-margin between the score of the ground-truth $y_i$ and the score of all other predictions. The above problem (3) is a Quadratic Program (QP) with $n|\mathcal{Y}|$ constraints, which is typically exponentially large. If an efficient separation oracle for identifying the most violated constraint is available, then a cutting-plane approach can be used to solve the QP. A cutting-plane algorithm maintains a working set of constraints and incrementally adds the most violated constraint to this working set while solving for the optimum solution under the working set. Tsochantaridis *et al*. [28] showed that such a procedure converges in a polynomial number of steps.

# 3  Multiple-Output Structured-Output Prediction

We now describe our proposed formulation for multiple-output structured-output prediction.

**Model.** Our model is a generalization of the single-output SSVM. A multiple-output SSVM is a mapping from the input space $\mathcal{X}$ to an $M$-tuple[1] of structured outputs $Y_i = \{\hat{y}_i^1, \ldots, \hat{y}_i^M \mid \hat{y}_i \in \mathcal{Y}\}$, given by $g : \mathcal{X} \to \mathcal{Y}^M$, where $g(x) = \operatorname{argmax}_{Y \in \mathcal{Y}^M} \mathbf{W}^T \boldsymbol{\Phi}(x, Y)$. Notice that the joint feature map is now a function of the input and the entire *set* of predicted structured-outputs, *i.e.*, $\boldsymbol{\Phi} : \mathcal{X} \times \mathcal{Y}^M \to \mathbb{R}^d$. Without further assumptions, optimizing over the output space $|\mathcal{Y}|^M$ would be intractable. We make a mean-field-like simplifying assumption that the set score *factors* into independent predictor scores, *i.e.*, $\boldsymbol{\Phi}(x_i, Y) = [\, \boldsymbol{\phi}^1(x_i, y^1)^T, \ldots, \boldsymbol{\phi}^M(x_i, y^M)^T \,]^T$. Thus, $g$ is composed of $M$ single-output predictors: $g(x) = \left( f^1(x), \ldots, f^M(x) \right)$, where $f^m(x) = \operatorname{argmax}_{y \in \mathcal{Y}} \mathbf{w}_m^T \boldsymbol{\phi}^m(x, y)$. Hence, the multiple-output SSVM is parameterized by an $M$-tuple of weight vectors: $\mathbf{W} = [\mathbf{w}_1^T, \ldots, \mathbf{w}_M^T]^T$.

## 3.1  Multiple-Output Loss

Let $\hat{Y}_i = \{\hat{y}_i^1, \ldots, \hat{y}_i^M\}$ be the set of predicted outputs for input $x_i$, *i.e.*, $\hat{y}_i^m = f^m(x_i)$. In the single-output SSVM, there typically exists a ground-truth output $y_i$ for each datapoint, and the quality of $\hat{y}_i$ w.r.t. $y_i$ is given by $\ell(y_i, \hat{y}_i)$.

**How good is a set of outputs?** For our multiple-output predictor, we need to define a task-specific loss function that can measure the quality of any set of predictions $\hat{Y}_i \in \mathcal{Y}^M$. Ideally, the quality of these predictions should be evaluated by the secondary mechanism that uses these predictions. For instance, in an interactive setting where they are shown to a user, the quality of $\hat{Y}_i$ could be measured by how much it reduces the user-interaction time. In the M-best hypotheses re-ranking scenario, the accuracy of the top single output *after* re-ranking could be used as the quality measure for $\hat{Y}_i$. While multiple options exist, in order to provide a general formulation and to isolate our approach, we propose the "oracle" or "hindsight" set-loss as a surrogate:

$$\mathcal{L}(\hat{Y}_i) = \min_{\hat{y}_i \in \hat{Y}_i} \ell(y_i, \hat{y}_i) \tag{4}$$

*i.e.*, the set of predictions $\hat{Y}_i$ only pays a loss for the *most accurate* prediction contained in this set (*e.g.*, the best segmentation of an image, or the best translation of a sentence). This loss has the desirable behaviour that predicting a set that contains even a single accurate output is better than predicting a set that has none. Moreover, only being penalized for the most accurate prediction allows an ensemble to hedge its bets without having to pay for being too diverse (this is opposite to the effect that replacing $\min$ with $\max$ or avg. would have). However, this also makes the set-loss rather poorly conditioned – if even a single prediction in the ensemble is the ground-truth, the set-loss is 0, no matter what else is predicted.

**Hinge-like Upper-Bound.** The set-loss $\mathcal{L}(\hat{Y}_i(\mathbf{W}))$ is a non-continuous non-convex function of $\mathbf{W}$ and is thus difficult to optimize. If unique ground-truth sets $Y_i$ were available, we could set up a standard hinge-loss approximation:

$$H_i(\mathbf{W}) = \max_{Y \in \mathcal{Y}^M} \left( \mathcal{L}(Y) + \mathbf{W}^T \mathbf{\Phi}(x_i, Y) \right) - \mathbf{W}^T \mathbf{\Phi}(x_i, Y_i) \tag{5}$$

where $\mathbf{\Phi}(x_i, Y) = [\,\boldsymbol{\phi}^1(x_i, y^1)^T, \ldots, \boldsymbol{\phi}^M(x_i, y^M)^T\,]^T$ are stacked joint feature maps.

However, no such natural choice for $Y_i$ exists. We propose a hinge-like upper-bound on the set-loss, that we refer to as min-hinge:

$$\tilde{H}_i(\mathbf{W}) = \min_{m \in [M]} \hbar_i(\mathbf{w}_m), \tag{6}$$

*i.e.*, we take the $\min$ over the hinge-losses (2) corresponding to each of the $M$ predictors. Since each hinge-loss is an upper-bound on the corresponding task-loss, *i.e.*, $\hbar_i(\mathbf{w}_m) \geq \ell(y_i, f^m(x_i))$, it is straightforward to see that the min-hinge is an upper-bound on the set-loss, *i.e.*, $\tilde{H}_i(\mathbf{W}) \geq \mathcal{L}(\hat{Y}_i)$. Notice that min-hinge is a $\min$ of convex functions, and thus not guaranteed to be convex.

## 3.2 Coordinate Descent for Learning Multiple Predictors

We now present our algorithm for learning a multiple-output SSVM by minimizing the regularized min-hinge loss:

$$\min_{\mathbf{W}} \quad \frac{1}{2} \|\mathbf{W}\|_2^2 + C \sum_{i \in [n]} \tilde{H}_i(\mathbf{W}) \tag{7}$$

We begin by rewriting the min-hinge loss in terms of indicator "flag" variables, *i.e.*,

$$\min_{\mathbf{W}, \{\rho_{i,m}\}} \quad \frac{1}{2} \|\mathbf{W}\|_2^2 + C \sum_{i \in [n]} \sum_{m \in [M]} \rho_{i,m}\, \hbar_i(\mathbf{w}_m) \tag{8a}$$

$$s.t. \quad \sum_{m \in [M]} \rho_{i,m} = 1 \qquad\qquad \forall i \in [n] \tag{8b}$$

$$\rho_{i,m} \in \{0, 1\} \qquad\qquad \forall i \in [n],\, m \in [M] \tag{8c}$$

where $\rho_{i,m}$ is a flag variable that indicates which predictor produces the smallest hinge-loss.

Optimization problem 8 is a mixed-integer quadratic programming problem (MIQP), which is NP-hard in general. However, we can exploit the structure of the problem via a block-coordinate descent algorithm where $\mathbf{W}$ and $\{\rho_{i,m}\}$ are optimized iteratively:

1. **Fix W; Optimize all $\{\rho_{i,m}\}$.**
   Given $\mathbf{W}$, the optimization over $\{\rho_{i,m}\}$ reduces to the minimization of $\sum_{i \in [n]} \sum_{m \in [M]} \rho_{i,m}\, \hbar_i(\mathbf{w}_m)$ subject to the "pick-one-predictor" constraints (8b, 8c). This decomposes into $n$ independent problems, which simply identify the best predictor for each datapoint according to the current hinge-losses, *i.e.*:

$$\rho_{i,m} = \begin{cases} 1 & \text{if} \quad m = \underset{m \in [M]}{\operatorname{argmin}} \hbar_i(\mathbf{w}_m) \\ 0 & \text{else.} \end{cases} \tag{9}$$

2. **Fix $\{\rho_{i,m}\}$; Optimize W.**
   Given $\{\rho_{i,m}\}$, optimization over $\mathbf{W}$ decomposes into $M$ independent problems, one for each predictor, which are equivalent to single-output SSVM learning problems:

$$\min_{\mathbf{W}} \ \frac{1}{2} \|\mathbf{W}\|_2^2 + C \sum_{i \in [n]} \sum_{m \in [M]} \rho_{i,m} \ \hbar_i(\mathbf{w}_m) \tag{10a}$$

$$= \min_{\mathbf{W}} \ \sum_{m \in [M]} \left\{ \frac{1}{2} \|\mathbf{w}_m\|_2^2 + C \sum_{i \in [n]} \rho_{i,m} \ \hbar_i(\mathbf{w}_m) \right\} \tag{10b}$$

$$= \sum_{m \in [M]} \min_{\mathbf{w}_m} \ \left\{ \frac{1}{2} \|\mathbf{w}_m\|_2^2 + C \sum_{i : \rho_{i,m} \neq 0} \hbar_i(\mathbf{w}_m) \right\} \tag{10c}$$

Thus, each subproblem in 10c can be optimized using using any standard technique for training SSVMs. We use the 1-slack algorithm of [14].

**Convergence.** Overall, the block-coordinate descent algorithm above iteratively assigns each data-point to a particular predictor (Step 1) and then independently trains each predictor with just the points that were assigned to it (Step 2). This is fairly reminiscent of k-means, where step 1 can be thought of as the member re-assignment step (or the M-step in EM) and step 2 can be thought of as the cluster-fitting step (or the E-step in EM). Since the flag variables take on discrete values and the objective function is non-increasing with iterations, the algorithm is guaranteed to converge in a finite number of steps.

**Generalization.** Formulation (8) can be generalized by replacing the "pick-one-predictor" constraint with "pick-K-predictors", *i.e.*, $\sum_{m \in [M]} \rho_{i,m} = K$, where $K$ is a robustness parameter that allows training data overlap between predictors. The M-step (cluster reassignment) is still simple, and involves assigning a data-point to the top K best predictors. The E-step is unchanged. Notice that at $K = M$, all predictors learn the same mapping. We analyze the effect of K in our experiments.

## 4 Related Work

At first glance, our work seems related to the multi-label classification literature, where the goal is to predict multiple labels for each input instance (*e.g.*, text tags for images on Flickr). However, the motivation and context of our work is fundamentally different. Specifically, in multi-label classification there are multiple possible labels for each instance and the goal is to predict as many of them as possible. On the other hand, in our setting there is a single ground-truth label for each instance and the learner makes multiple guesses, all of which are evaluated against that single ground-truth.

For the unstructured setting (*i.e.* when $|\mathcal{Y}|$ is polynomial), Dey *et al*. [9] proposed an algorithm that learns a multi-class classifier for each "slot" in a $M$-Best list, and provide a formal regret reduction from submodular sequence optimization.

To the best of our knowledge, the only other work that explicitly addresses the task of predicting multiple *structured* outputs is multi-label structured prediction (MLSP) [19]. This work may be seen as a technique to output predictions in the power-set of $\mathcal{Y}$ ($2^{\mathcal{Y}}$) with a learning cost comparable to algorithms for prediction over $\mathcal{Y}$. Most critically, MLSP requires gold-standard *sets* of labels (one set for each training example). In contrast, MCL neither needs nor has access to gold-standard sets. At a high-level, MCL and MLSP are orthogonal approaches, *e.g.*, we could introduce MLSP within MCL to create an algorithm that predicts multiple (diverse) sets of structured-outputs (e.g., multiple guesses by the algorithm where *each* guess is a *set* of bounding boxes of objects in an image).

A form of min-set-loss has received some attention in the context of ambiguously or incompletely annotated data. For instance, [4] trains an SSVM for object detection implicitly defining a task-adapted loss, $\mathcal{L}^{\min}(Y, \hat{y}) = \min_{y \in Y} \ell(y, \hat{y})$. Note that in this case there is a set of ground-truth labels and the model's prediction is a single label (evaluated against the closest ground-truth).

Our formulation is also reminiscent of a Latent-SSVM with the indicator flags $\{\rho_{i,m} \mid m \in [M]\}$ taking a role similar to latent variables. However, the two play very different roles. Latent variable models typically maximize or marginalize the model score across the latent variables, while MCL uses the flag variables as a representation of the oracle loss.

At a high-level, our ideas are also related to ensemble methods [21] like boosting. However, the key difference is that ensemble methods attempt to combine outputs from multiple weak predictors to ultimately make a single prediction. We are interested in making multiple predictions which will *all*

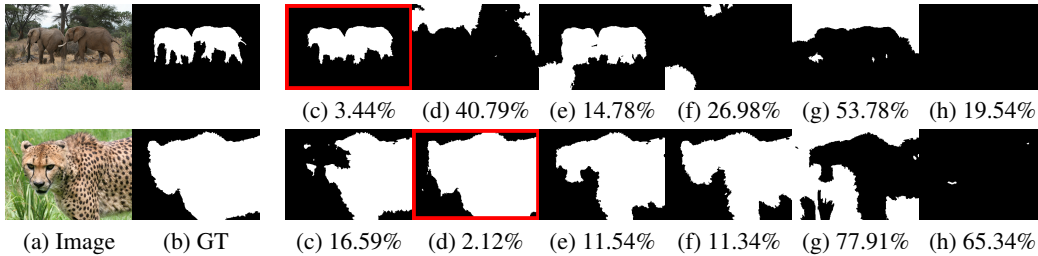

(c) 3.44%   (d) 40.79%   (e) 14.78%   (f) 26.98%   (g) 53.78%   (h) 19.54%

(a) Image   (b) GT   (c) 16.59%   (d) 2.12%   (e) 11.54%   (f) 11.34%   (g) 77.91%   (h) 65.34%

Figure 1: Each row shows the: (a) input image (b) ground-truth segmentation and (c-h) the set of predictions produced by MCL ($M = 6$). Red border indicates the most accurate segmentation (*i.e.*, lowest error). We can see that the predictors produce different plausible foreground hypotheses, *e.g.*, predictor (g) thinks foliage-like things are foreground.

be handed to an expert or secondary mechanism that has access to more complex (*e.g.*, higher-order) features.

# 5    Experiments

**Setup.** We tested algorithm MCL on two problems: i) foreground-background segmentation in image collections and ii) protein side-chain prediction. In both problems making a single perfect prediction is difficult due to inherent ambiguity in the tasks. Moreover, inference-time computing limitations force us to learn restricted models (*e.g.*, pairwise attractive CRFs) that may never be able to capture the true solution with a single prediction. The goal of our experiments is to study how much predicting a set of plausible hypotheses helps. Our experiments will show that MCL is able to produce sets of hypotheses which contain more accurate predictions than other algorithms and baselines aimed at producing multiple hypotheses.

## 5.1    Foreground-Background Segmentation

**Dataset.** We used the co-segmentation dataset, iCoseg, of Batra *et al*. [2]. iCoseg consists of 37 groups of related images mimicking typical consumer photograph collections. Each group may be thought of as an "event" (*e.g.*, images from a baseball game, a safari, *etc*.). The dataset provides pixel-level ground-truth foreground-background segmentations for each image. We used 9 difficult groups from iCoseg containing 166 images in total. These images were then split into train, validation and test sets of roughly equal size. See Fig. 1, 2 for some example images and segmentations.

**Model and Features.** The segmentation task is modeled as a binary pairwise MRF where each node corresponds to a superpixel [1] in the image. We extracted 12-dim color features at each superpixel (mean RGB; mean HSV; 5 bin Hue histogram; Hue histogram entropy). The edge features, computed for each pair of adjacent superpixels, correspond to a standard Potts model and a contrast sensitive Potts model. The weights at each edge were constrained to be positive so that the resulting supermodular potentials could be maximized via graph-cuts [6, 17].

**Baselines and Evaluation.** We compare our algorithm against three alternatives for producing multiple predictions: i) Single SSVM + $M$-Best MAP [29], ii) Single SSVM + Diverse $M$-Best MAP [3] and iii) Clustering + Multiple SSVMs.

For the first two baselines, we used all training images to learn a single SSVM and then produced multiple segmentations via $M$-Best MAP and Diverse $M$-Best MAP [3]. The $M$-Best MAP baseline was implemented via the BMMF algorithm [29] using dynamic graph-cuts [15] for computing max-marginals efficiently. For the Diverse $M$-Best MAP baseline we implemented the DIVMBEST algorithm of Batra *et al*. [3] using dynamic graph-cuts. The third baseline, Clustering + Multiple SSVM (C-SSVM), involves first clustering the training images into $M$ clusters and then training $M$ SSVMs independently on each cluster. For clustering, we used k-means with $\ell_2$ distance on color features (same as above) computed on foreground pixels.

For each algorithm we varied the number of predictors $M \in \{1, 2, \ldots, 6\}$ and tuned the regularization parameter $C$ on validation. Since MCL involves non-convex optimization, a good initialization is important. We used the output of k-means clustering as the initial assignment of images to predictors, so MCL's first coordinate descent iteration produces the same results as C-SSVM. The task-loss

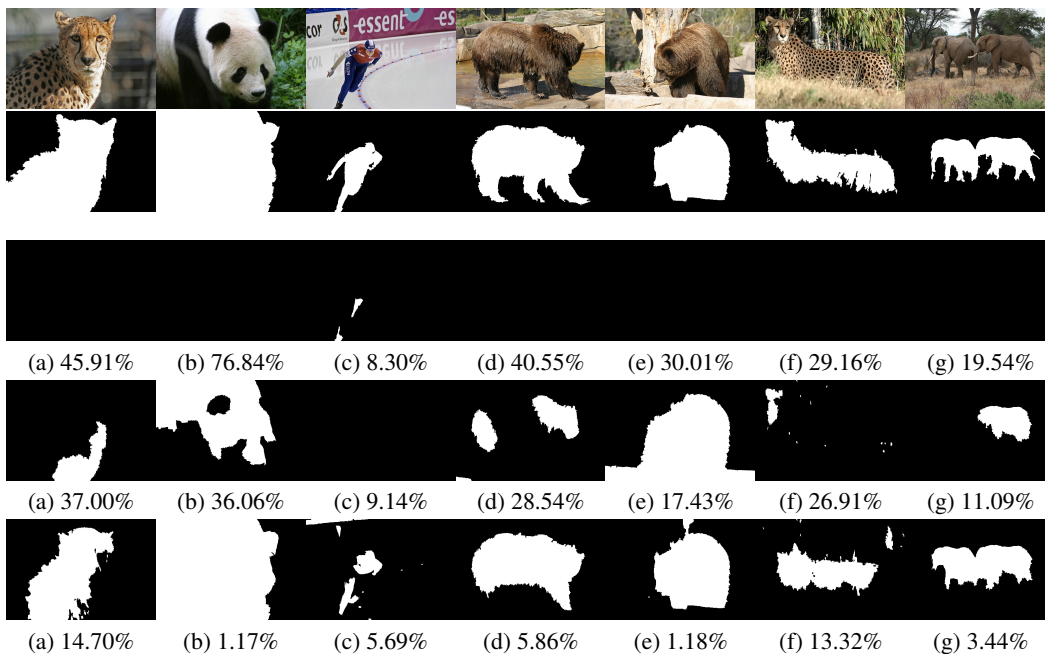

Figure 2: In each column: first row shows input images; second shows ground-truth; third shows segmentation produced by the single SSVM baseline; and the last two rows show the best MCL predictions ($M = 6$) at the end of the first and last coordinate descent iteration.

in this experiment ($\ell$) is the percentage of incorrectly labeled pixels, and the evaluation metric is the set-loss, $\mathcal{L} = \min_{\hat{y}_i \in \hat{Y}_i} \ell(y_i, \hat{y}_i)$, *i.e.*, the pixel error of the best segmentation among all predictions.

**Comparison against Baselines.** Fig. 3a show the performance of various algorithms as a function of the number of predictors $M$. We observed that $M$-Best MAP produces nearly identical predictions and thus the error drops negligibly as $M$ is increased. On the other hand, the diverse $M$-Best predictions output by DIVMBEST [3] lead to a substantial drop in the set-loss. MCL outperforms both DIVMBEST and C-SSVM, confirming our hypothesis that it is beneficial to learn a collection of predictors, rather than learning a single predictor and making diverse predictions from it.

**Behaviour of Coordinate Descent.** Fig. 3b shows the MCL objective and train/test errors as a function of the coordinate descent steps. We verify that the objective function is improved at every iteration and notice a nice correlation between the objective and the train/test errors.

**Effect of $C$.** Fig. 3c compares performance for different values of regularization parameter $C$. We observe a fairly stable trend with MCL consistently outperforming baselines.

**Effect of $K$.** Fig. 3d shows the performance of MCL as robustness parameter $K$ is increased from 1 to $M$. We observe a monotonic reduction in error as $K$ decreases, which suggests there is a natural clustering of the data and thus learning a single SSVM is detrimental.

**Qualitative Results.** Fig. 1 shows example images, ground-truth segmentations, and the predictions made by $M = 6$ predictors. We observe that the $M$ hypotheses are both diverse and plausible. The evolution of the best prediction with coordinate descent iterations can be seen in Fig. 2.

## 5.2 Protein Side-Chain Prediction

**Model and Dataset.** Given a protein backbone structure, the task here is to predict the amino acid side-chain configurations. This problem has been traditionally formulated as a pairwise MRF with node labels corresponding to (discretized) side-chain configurations (*rotamers*). These models include pairwise interactions between nearby side-chains, and between side-chains and backbone. We use the dataset of [7] which consists of 276 proteins (up to 700 residues long) split into train and test sets of sizes 55 and 221 respectively.[2] The energy function is defined as a weighted sum of eight

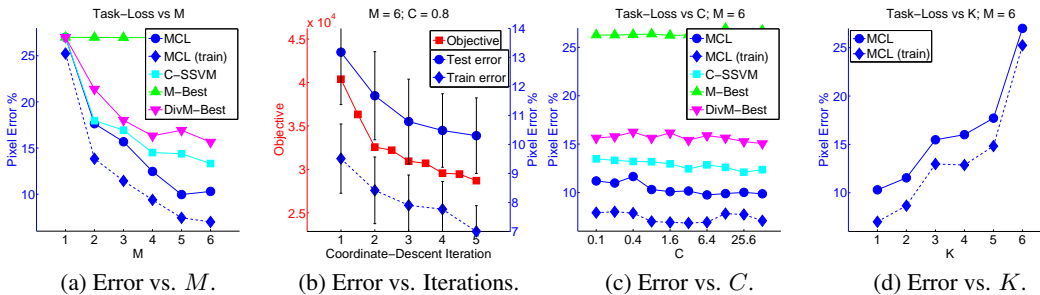

(a) Error vs. $M$.  (b) Error vs. Iterations.  (c) Error vs. $C$.  (d) Error vs. $K$.

Figure 3: Experiments on foreground-background segmentation.

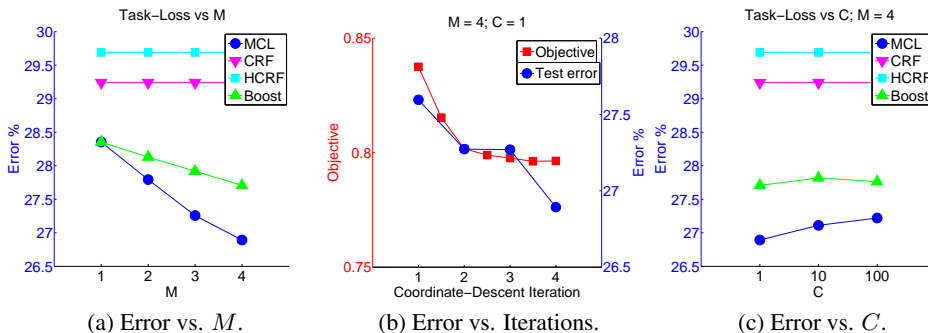

(a) Error vs. $M$.  (b) Error vs. Iterations.  (c) Error vs. $C$.

Figure 4: Experiments on protein side-chain prediction.

known energy terms where the weights are to be learned. We used TRW-S [16] (early iterations) and ILP (CPLEX [13]) for inference.

**Baselines and Evaluation.** For this application there is no natural analogue to the C-SSVM baseline and thus we used a boosting-like baseline where we first train an SSVM on the entire training data; use the training instances with high error to train a second SSVM, and so on. For comparison, we also report results from the CRF and HCRF models proposed in [7]. Following [7], we report average error rates for the first two angles ($\chi_1$ and $\chi_2$) on all test proteins.

**Results.** Fig. 4 shows the results. Overall, we observe behavior similar to the previous set of experiments. Fig. 4a confirms that multiple predictors are beneficial, and that MCL is able to outperform the boosting-like baseline. Fig. 4b shows the progress of the MCL objective and test loss with coordinate descent iterations; we again observe a positive correlation between the objective and the loss. Fig. 4c shows that MCL outperforms baselines across a range of values of $C$.

## 6 Discussion and Conclusions

We presented an algorithm for producing a set of structured outputs and argued that in a number of problems it is beneficial to generate a set of plausible and diverse hypotheses. Typically, this is accomplished by learning a single-output model and then producing $M$-best hypotheses from it. This causes a disparity between the way the model is trained (to produce a single output) and the way it is used (to produce multiple outputs). Our proposed algorithm (MCL) provides a principled way to directly optimize the multiple prediction min-set-loss.

There are a number of directions to extend this work. While we evaluated performance of all algorithms in terms of oracle set-loss, it would be interesting to measure the impact of MCL and other baselines on user experience or final stage performance in cascaded algorithms.

Further, our model assumes a modular scoring function $S(Y) = \mathbf{W}^T \Phi(x, Y) = \sum_{m \in [M]} \mathbf{w}_m^T \phi^m(x, y^m)$, *i.e.*, the score of a set is the sum of the scores of its members. In a number of situations, the score $S(Y)$ might be a *submodular* function. Such scoring functions often arise when we want the model to explicitly reward diverse subsets. We plan to make connections with greedy-algorithms for submodular maximization for such cases.

**Acknowledgments:** We thank David Sontag for his assistance with the protein data. AGR was supported by the C2S2 Focus Center (under the SRC's Focus Center Research Program).

## Footnotes

[1]Our formulation is described with a *nominal* ordering of the predictions. However, both the proposed objective function and optimization algorithm are invariant to permutations of this ordering.

[2]Dataset available from: http://cyanover.fhcrc.org/recomb-2007/

# References

[1] R. Achanta, A. Shaji, K. Smith, A. Lucchi, P. Fua, and S. Ssstrunk. SLIC Superpixels Compared to State-of-the-art Superpixel Methods. *PAMI*, (To Appear) 2012. 6

[2] D. Batra, A. Kowdle, D. Parikh, J. Luo, and T. Chen. iCoseg: Interactive Co-segmentation with Intelligent Scribble Guidance. In *CVPR*, 2010. 6

[3] D. Batra, P. Yadollahpour, A. Guzman-Rivera, and G. Shakhnarovich. Diverse M-Best Solutions in Markov Random Fields. In *ECCV*, 2012. 2, 6, 7

[4] M. B. Blaschko and C. H. Lampert. Learning to Localize Objects with Structured Output Regression. In *ECCV*, 2008. 5

[5] Y. Boykov and M.-P. Jolly. Interactive Graph Cuts for Optimal Boundary and Region Segmentation of Objects in N-D Images. *ICCV*, 2001. 1

[6] Y. Boykov, O. Veksler, and R. Zabih. Efficient Approximate Energy Minimization via Graph Cuts. *PAMI*, 20(12):1222–1239, 2001. 6

[7] O. S.-F. Chen Yanover and Y. Weiss. Minimizing and Learning Energy Functions for Side-Chain Prediction. *Journal of Computational Biology*, 15(7):899–911, 2008. 7, 8

[8] M. Collins. Discriminative Reranking for Natural Language Parsing. In *ICML*, pages 175–182, 2000. 2

[9] D. Dey, T. Y. Liu, M. Hebert, and J. A. Bagnell. Contextual sequence prediction with application to control library optimization. In *Robotics: Science and Systems*, 2012. 5

[10] M. Everingham, L. Van Gool, C. K. I. Williams, J. Winn, and A. Zisserman. The PASCAL Visual Object Classes Challenge 2011 (VOC2011) Results. http://www.pascal-network.org/challenges/VOC/voc2011/workshop/index.html. 2

[11] M. Fromer and A. Globerson. An LP View of the M-best MAP problem. In *NIPS*, 2009. 2

[12] L. Huang and D. Chiang. Better K-best parsing. In *Proceedings of the Ninth International Workshop on Parsing Technology (IWPT)*, pages 53–64, 2005. 2

[13] IBM Corporation. IBM ILOG CPLEX Optimization Studio. http://www-01.ibm.com/software/integration/optimization/cplex-optimization-studio/, 2012. 8

[14] T. Joachims, T. Finley, and C.-N. Yu. Cutting-Plane Training of Structural SVMs. *Machine Learning*, 77(1):27–59, 2009. 5

[15] P. Kohli and P. H. S. Torr. Measuring Uncertainty in Graph Cut Solutions. *CVIU*, 112(1):30–38, 2008. 6

[16] V. Kolmogorov. Convergent Tree-Reweighted Message Passing for Energy Minimization. *PAMI*, 28(10):1568–1583, 2006. 8

[17] V. Kolmogorov and R. Zabih. What Energy Functions can be Minimized via Graph Cuts? *PAMI*, 26(2):147–159, 2004. 6

[18] J. D. Lafferty, A. McCallum, and F. C. N. Pereira. Conditional Random Fields: Probabilistic Models for Segmenting and Labeling Sequence Data. In *ICML*, 2001. 1

[19] C. H. Lampert. Maximum Margin Multi-Label Structured Prediction. In *NIPS*, 2011. 5

[20] E. L. Lawler. A Procedure for Computing the K Best Solutions to Discrete Optimization Problems and Its Application to the Shortest Path Problem. *Management Science*, 18:401–405, 1972. 2

[21] D. W. Opitz and R. Maclin. Popular Ensemble Methods: An Empirical Study. *J. Artif. Intell. Res. (JAIR)*, 11:169–198, 1999. 5

[22] K. Papineni, S. Roukos, T. Ward, and W.-J. Zhu. BLEU: a Method for Automatic Evaluation of Machine Translation. In *ACL*, 2002. 2

[23] D. Park and D. Ramanan. N-Best Maximal Decoders for Part Models. In *ICCV*, 2011. 2

[24] A. Pauls, D. Klein, and C. Quirk. Top-Down K-Best A* Parsing. In *ACL*, 2010. 2

[25] C. Rother, V. Kolmogorov, and A. Blake. "GrabCut" – Interactive Foreground Extraction using Iterated Graph Cuts. *SIGGRAPH*, 2004. 1

[26] L. Shen, A. Sarkar, and F. J. Och. Discriminative Reranking for Machine Translation. In *HLT-NAACL*, pages 177–184, 2004. 2

[27] B. Taskar, C. Guestrin, and D. Koller. Max-Margin Markov Networks. In *NIPS*, 2003. 1

[28] I. Tsochantaridis, T. Joachims, T. Hofmann, and Y. Altun. Large Margin Methods for Structured and Interdependent Output Variables. *JMLR*, 6:1453–1484, 2005. 1, 2, 3

[29] C. Yanover and Y. Weiss. Finding the M Most Probable Configurations Using Loopy Belief Propagation. In *NIPS*, 2003. 2, 6

